# Context-Dependent Multiple Distribution Phonetic Modeling with MLPs

**Michael Cohen**
SRI International
Menlo Park, CA 94025

**Horacio Franco**
SRI International

**Nelson Morgan**
Intl. Computer Science Inst.
Berkeley, CA 94704

**David Rumelhart**
Stanford University
Stanford, CA 94305

**Victor Abrash**
SRI International

## Abstract

A number of hybrid multilayer perceptron (MLP)/hidden Markov model (HMM) speech recognition systems have been developed in recent years (Morgan and Bourlard, 1990). In this paper, we present a new MLP architecture and training algorithm which allows the modeling of context-dependent phonetic classes in a hybrid MLP/HMM framework. The new training procedure smooths MLPs trained at different degrees of context dependence in order to obtain a robust estimate of the context-dependent probabilities. Tests with the DARPA Resource Management database have shown substantial advantages of the context-dependent MLPs over earlier context-independent MLPs, and have shown substantial advantages of this hybrid approach over a pure HMM approach.

## 1 INTRODUCTION

Hidden Markov models are used in most current state-of-the-art continuous-speech recognition systems. A hidden Markov model (HMM) is a stochastic finite state machine with two sets of probability distributions. Associated with each state is a probability distribution over transitions to next states and a probability distribution over output symbols (often referred to as observation probabilities). When applied to continuous speech, the observation probabilities are typically used to model local speech features such as spectra, and the transition probabilities are used to model the displacement of these features through time. HMMs of individual phonetic segments (phones) can be concatenated to model words and word models can be concatenated, according to a grammar, to model sentences, resulting in a finite state representation of acoustic-phonetic, phonological, and syntactic structure.

The HMM approach is limited by the need for strong statistical assumptions that are unlikely to be valid for speech. Previous work by Morgan and Bourlard (1990) has shown both theoretically and practically that some of these limitations can be overcome by using multilayer perceptrons (MLPs) to estimate the HMM state-dependent observation probabilities. In addition to relaxing the restrictive independence assumptions of traditional HMMs, this approach results in a reduction in the number of parameters needed for detailed phonetic modeling as a result of increased sharing of model parameters between phonetic classes.

Recently, this approach was applied to the SRI-DECIPHER™ system, a state-of-the-art continuous speech recognition system (Cohen et al., 1990), using an MLP to provide estimates of context-independent posterior probabilities of phone classes, which were then converted to HMM context-independent state observation likelihoods using Bayes' rule (Renals et al., 1992). In this paper, we describe refinements of the system to model phonetic classes with a sequence of context-dependent probabilities.

*Context-dependent modeling:* The realization of individual phones in continuous speech is highly dependent upon phonetic context. For example, the sound of the vowel /ae/ in the words "map" and "tap" is different, due to the influence of the preceding phone. These context effects are referred to as "coarticulation". Experience with HMM technology has shown that using context-dependent phonetic models improves recognition accuracy significantly (Schwartz et al., 1985). This is so because acoustic correlates of coarticulatory effects are explicitly modeled, producing sharper and less overlapping probability density functions for the different phone classes.

Context-dependent HMMs use different probability distributions for every phone in every different relevant context. This practice causes problems that are due to the reduced amount of data available to train phones in highly specific contexts, resulting in models that are not robust and generalize poorly. The solution to this problem used by many HMM systems is to train models at many different levels of context-specificity, including biphone (conditioned only on the phone immediately to the left or right), generalized biphone (conditioned on the broad class of the phone to the left or right), triphone (conditioned on the phone to the left and the right), generalized triphone, and word specific phone. Models conditioned by more specific contexts are linearly smoothed with more general models. The "deleted interpolation" algorithm (Jelinek and Mercer, 1980) provides linear weighting coefficients for the observation probabilities with different degrees of context dependence by maximizing the likelihood of the different models over new, unseen data. This approach cannot be directly extended to MLP-based systems because averaging the weights of two MLPs does not result in an MLP with the average performance. It would be possible to use this approach to average the probabilities that are output from different MLPs; however, since the MLP training algorithm is a discriminant procedure, it would be desirable to use a discriminant or error-based procedure to smooth the MLP probabilities together.

An earlier approach to context-dependent phonetic modeling with MLPs was proposed by Bourlard et al. (1992). It is based on factoring the context-dependent likelihood and uses a set of binary inputs to the network to specify context classes. The number

of parameters and the computational load using this approach are not much greater than those for the original context-independent net.

The context-dependent modeling approach we present here uses a different factoring of the desired context-dependent likelihoods, a network architecture that shares the input-to-hidden layer among the context-dependent classes to reduce the number of parameters, and a training procedure that smooths networks with different degrees of context-dependence in order to achieve robustness in probability estimates.

*Multidistribution modeling:* Experience with HMM-based systems has shown the importance of modeling phonetic units with a sequence of distributions rather than a single distribution. This allows the model to capture some of the dynamics of phonetic segments. The SRI-DECIPHER™ system models most phones with a sequence of three HMM states. Our initial hybrid system used only a single MLP output unit for each HMM phonetic class. This output unit supplied the probability for all the states of the associated phone model.

Our initial attempt to extend the hybrid system to the modeling of a sequence of distributions for each phone involved increasing the number of output units from 69 (corresponding to phone classes) to 200 (corresponding to the states of the HMM phone models). This resulted in an increase in word-recognition error rate by almost 30%. Experiments at ICSI had a similar result (personal communication). The higher error rate seemed to be due to the discriminative nature of the MLP training algorithm. The new MLP, with 200 output units, was attempting to discriminate sub-phonetic classes, corresponding to HMM states. As a result, the MLP was attempting to discriminate into separate classes acoustic vectors that corresponded to the same phone and, in many cases, were very similar but were aligned with different HMM states. There were likely to have been many cases in which almost identical acoustic training vectors were labeled as a positive example in one instance and a negative example in another for the same output class. The appropriate level at which to train discrimination is likely to be the level of the phone (or higher) rather than the sub-phonetic HMM-state level (to which these outputs units correspond). The new architecture presented here accomplishes this by training separate output layers for each of the three HMM states, resulting in a network trained to discriminate at the phone level, while allowing three distributions to model each phone. This approach is combined with the context-dependent modeling approach, described in Section 3.

## 2 HYBRID MLP/HMM

The SRI-DECIPHER™ system is a phone-based, speaker-independent, continuous-speech recognition system, based on semicontinuous (tied Gaussian mixture) HMMs (Cohen et al., 1990). The system extracts four features from the input speech waveform, including 12th-order mel cepstrum, log energy, and their smoothed derivatives. The front end produces the 26 coefficients for these four features for each 10-ms frame of speech.

Training of the phonetic models is based on maximum-likelihood estimation using the forward-backward algorithm (Levinson et al., 1983). Recognition uses the Viterbi algorithm (Levinson et al., 1983) to find the HMM state sequence (corresponding to a sentence) with the highest probability of generating the observed acoustic sequence.

The hybrid MLP/HMM DECIPHER™ system substitutes (scaled) probability estimates computed with MLPs for the tied-mixture HMM state-dependent observation

probability densities. No changes are made in the topology of the HMM system.

The initial hybrid system used an MLP to compute context-independent phonetic probabilities for the 69 phone classes in the DECIPHER™ system. Separate probabilities were not computed for the different states of phone models. During the Viterbi recognition search, the probability of acoustic vector $Y_t$ given the phone class $q_j$, $P(Y_t|q_j)$, is required for each HMM state. Since MLPs can compute Bayesian posterior probabilities, we compute the required HMM probabilities using

$$P(Y_t|q_j) = \frac{P(q_j|Y_t)P(Y_t)}{P(q_j)} \tag{1}$$

The factor $P(q_j|Y_t)$ is the posterior probability of phone class $q_j$ given the input vector $Y$ at time $t$. This is computed by a backpropagation-trained (Rumelhart et al., 1986) three-layer feed-forward MLP. $P(q_j)$ is the prior probability of phone class $q_j$ and is estimated by counting class occurrences in the examples used to train the MLP. $P(Y_t)$ is common to all states for any given time frame, and can therefore be discarded in the Viterbi computation, since it will not change the optimal state sequence used to get the recognized string.

The MLP has an input layer of 234 units, spanning 9 frames (with 26 coefficients for each) of cepstra, delta-cepstra, log-energy, and delta-log-energy that are normalized to have zero mean and unit variance. The hidden layer has 1000 units, and the output layer has 69 units, one for each context-independent phonetic class in the DECIPHER™ system. Both the hidden and output layers consist of sigmoidal units.

The MLP is trained to estimate $P(q_j|Y_t)$, where $q_j$ is the class associated with the middle frame of the input window. Stochastic gradient descent is used. The training signal is provided by the HMM DECIPHER™ system previously trained by the forward-backward algorithm. Forced Viterbi alignments (alignments to the known word string) for every training sentence provide phone labels, among 69 classes, for every frame of speech. The target distribution is defined as 1 for the index corresponding to the phone class label and 0 for the other classes. A minimum relative entropy between posterior target distribution and posterior output distribution is used as a training criterion. With this training criterion and target distribution, assuming enough parameters in the MLP, enough training data, and that the training does not get stuck in a local minimum, the MLP outputs will approximate the posterior class probabilities $P(q_j|Y_t)$ (Morgan and Bourlard, 1990). Frame classification on an independent cross-validation set is used to control the learning rate and to decide when to stop training as in Renals et al. (1992). The initial learning rate is kept constant until cross-validation performance increases less than 0.5%, after which it is reduced as $1/2^n$ until performance increases no further.

## 3 CONTEXT-DEPENDENCE

Our initial implementation of context-dependent MLPs models generalized biphone phonetic categories. We chose a set of eight left and eight right generalized biphone phonetic-context classes, based principally on place of articulation and acoustic characteristics. The context-dependent architecture is shown in Figure 1. A separate output layer (consisting of 69 output units corresponding to 69 context-dependent phonetic classes) is trained for each context. The context-dependent MLP can be viewed as a set of MLPs, one for each context, which have the same input-to-hidden

weights. Separate sets of context-dependent output layers are used to model context effects in different states of HMM phone models, thereby combining the modeling of multiple phonetic distributions and context-dependence. During training and recognition, speech frames aligned with first states of HMM phones are associated with the appropriate left context output layer, those aligned with last states of HMM phones are associated with the appropriate right context output layer, and middle states of three state models are associated with the context-independent output layer. As a result, since the training proceeds (as before) as if each output layer were part of an independent net, the system learns discrimination between the different phonetic classes within an output layer (which now corresponds to a specific context and HMM-state position), but does not learn discrimination between occurrences of the same phone in different contexts or between the different states of the same HMM phone.

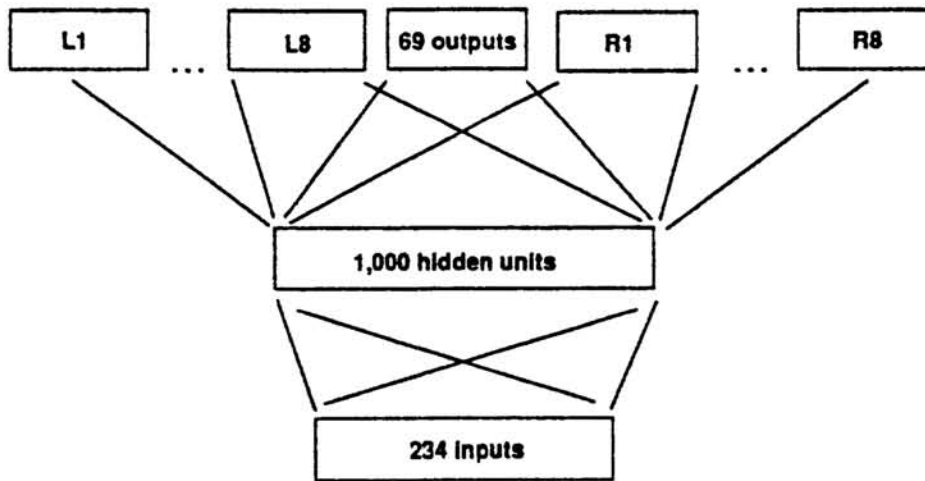

Figure 1: Context-Dependent MLP

## 3.1 CONTEXT-DEPENDENT FACTORING

In a context-dependent HMM, every state is associated with a specific phone class and context. During the Viterbi recognition search, $P(Y_t|q_j,c_k)$ (the probability of acoustic vector $Y_t$ given the phone class $q_j$ in the context class $c_k$) is required for each state. We compute the required HMM probabilities using

$$P(Y_t|q_j,c_k) = \frac{P(q_j|Y_t,c_k)P(Y_t|c_k)}{P(q_j|c_k)} \quad (2)$$

where $P(Y_t|c_k)$ can be factored again as

$$P(Y_t|c_k) = \frac{P(c_k|Y_t)P(Y_t)}{P(c_k)} \quad (3)$$

The factor $P(q_j|Y_t,c_k)$ is the posterior probability of phone class $q_j$ given the input vector $Y_t$ and the context class $c_k$. To compute this factor, we consider the conditioning on $c_k$ in (2) as restricting the set of input vectors only to those produced in the context $c_k$. If $M$ is the number of context classes, this implementation uses a set of $M$ MLPs (all sharing the same input-to-hidden layer) similar to those used in the context-independent case except that each MLP is trained using only input-output examples obtained from the corresponding context, $c_k$.

Every context-specific net performs a simpler classification than in the context-independent case because within a context the acoustics corresponding to different phones have less overlap.

$P(c_k|Y_t)$ is computed by a second MLP. A three-layer feed-forward MLP is used which has 1000 hidden units and an output unit corresponding to each context class. $P(q_j|c_k)$ and $P(c_k)$ are estimated by counting over the training examples. Finally, $P(Y_t)$ is common to all states for any given time frame, and can therefore be discarded in the Viterbi computation, since it will not change the optimal state sequence used to get the recognized string.

## 3.2 CONTEXT-DEPENDENT TRAINING AND SMOOTHING

We use the following method to achieve robust training of context-specific nets:

An initial context-independent MLP is trained, as described in Section 2, to estimate the context-independent posterior probabilities over the N phone classes. After the context-independent training converges, the resulting weights are used to initialize the weights going to the context-specific output layers. Context-dependent training proceeds by backpropagating error only from the appropriate output layer for each training example. Otherwise, the training procedure is similar to that for the context-independent net, using stochastic gradient descent and a relative-entropy training criterion. Overall classification performance evaluated on an independent cross-validation set is used to determine the learning rate and stopping point. Only hidden-to-output weights are adjusted during context-dependent training. We can view the separate output layers as belonging to independent nets, each one trained on a non-overlapping subset of the original training data.

Every context-specific net would asymptotically converge to the context conditioned posteriors $P(q_j|Y_t,c_k)$ given enough training data and training iterations. As a result of the initialization, the net starts estimating $P(q_j|Y_t)$, and from that point it follows a trajectory in weight space, incrementally moving away from the context-independent parameters as long as classification performance on the cross-validation set improves. As a result, the net retains useful information from the context-independent initial conditions. In this way, we perform a type of nonlinear smoothing between the pure context-independent parameters and the pure context-dependent parameters.

## 4 EVALUATION

Training and recognition experiments were conducted using the speaker-independent, continuous-speech, DARPA Resource Management database. The vocabulary size is 998 words. Tests were run both with a word-pair (perplexity 60) grammar and with no grammar. The training set for the HMM system and for the MLP consisted of the 3990 sentences that make up the standard DARPA speaker-independent training set for the Resource Management task. The 600 sentences making up the Resource Management February 89 and October 89 test sets were used for cross-validation during both the context-independent and context-dependent MLP training, and for tuning HMM system parameters (e.g., word transition weight).

Table 1: Percent Word Error and Parameter Count with Word-Pair Grammar

|        | CIMLP | CDMLP | HMM   | MIXED |
|--------|-------|-------|-------|-------|
| Feb91  | 5.8   | 4.7   | 3.8   | 3.2   |
| Sep92a | 10.9  | 7.6   | 10.1  | 7.7   |
| Sep92b | 9.5   | 6.6   | 7.0   | 5.7   |
| # Parms| 300K  | 1400K | 5500K | 6100K |

Table 2: Percent Word Error with No Grammar

|        | CIMLP | CDMLP | HMM   | MIXED |
|--------|-------|-------|-------|-------|
| Feb91  | 24.7  | 18.4  | 19.3  | 15.9  |
| Sep92a | 31.5  | 27.1  | 29.2  | 25.4  |
| Sep92b | 30.9  | 24.9  | 26.6  | 21.5  |

Table 1 presents word recognition error and number of system parameters for four different versions of the system, for three different Resource Management test sets using the word-pair grammar. Table 2 presents word recognition error for the corresponding tests with no grammar (the number of system parameters are the same as those shown in Table 1).

Comparing context-independent MLP (CIMLP) to context-dependent MLP (CDMLP) shows improvements with CDMLP in all six tests, ranging from a 15% to 30% reduction in word error. The CDMLP system combines multiple-distribution modeling with the context-dependent modeling technique. The CDMLP system performs better than the context-dependent HMM (CDHMM) system in five out of the six tests.

The MIXED system uses a weighted mixture of the logs of state observation likelihoods provided by the CIMLP and the CDHMM (Renals et al., 1992). This system shows the best recognition performance so far achieved with the DECIPHER™ system on the Resource Management database. In all six tests, it performs significantly better than the pure CDHMM system.

## 5 DISCUSSION

The results shown in Tables 1 and 2 suggest that MLP estimation of HMM observation likelihoods can improve the performance of standard HMMs. These results also suggest that systems that use MLP-based probability estimation make more efficient use of their parameters than standard HMM systems. In standard HMMs, most of the parameters in the system are in the observation distributions associated with the individual states of phone models. MLPs use representations that are more distributed in nature, allowing more sharing of representational resources and better allocation of representational resources based on training. In addition, since MLPs are trained to discriminate between classes, they focus on modeling boundaries between classes rather than class internals.

One should keep in mind that the reduction in memory needs that may be attained by replacing HMM distributions with MLP-based estimates must be traded off against increased computational load during both training and recognition. The MLP computations during training and recognition are much larger than the corresponding Gaussian mixture computations for HMM systems.

The results also show that the context-dependent modeling approach presented here substantially improves performance over the earlier context-independent MLP. In addition, the context-dependent MLP performed better than the context-dependent HMM in five out of the six tests although the CDMLP is a far simpler system than the CDHMM, with approximately a factor of four fewer parameters and modeling of only generalized biphone phonetic contexts. The CDHMM uses a range of context-dependent models including generalized and specific biphone, triphone, and word-specific phone. The fact that context-dependent MLPs can perform as well or better than context-dependent HMMs while using less specific models suggests that they may be more vocabulary-independent, which is useful when porting systems to new tasks. In the near future we will test the CDMLP system on new vocabularies.

The MLP smoothing approach described here can be extended to the modeling of finer context classes. A hierarchy of context classes can be defined in which each context class at one level is included in a broader class at a higher level. The context-specific MLP at a given level in the hierarchy is initialized with the weights of a previously trained context-specific MLP at the next higher level, and then finer context training can proceed as described in Section 3.2.

The distributed representation used by MLPs is exploited in the context-dependent modeling approach by sharing the input-to-hidden layer weights between all context classes. This sharing substantially reduces the number of parameters to train and the amount of computation required during both training and recognition. In addition, we do not adjust the input-to-hidden weights during the context-dependent phase of training, assuming that the features provided by the hidden layer activations are relatively low level and are appropriate for context-dependent as well as context-independent modeling. The large decrease in cross-validation error observed going from context-independent to context-dependent MLPs (30.6% to 21.4%) suggests that the features learned by the hidden layer during the context-independent training phase, combined with the extra modeling power of the context-specific hidden-to-output layers, were adequate to capture the more detailed context-specific phone classes.

The best performance shown in Tables 1 and 2 is that of the MIXED system, which combines CIMLP and CDHMM probabilities. The CDMLP probabilities can also be combined with CDHMM probabilities; however, we hope that the planned extension of our CDMLP system to finer contexts will lead to a better system than the MIXED system without the need for such mixing, therefore resulting in a simpler system.

The context-dependent MLP shown here has more than 1,400,000 weights. We were able to robustly train such a large network by using a cross-validation set to determine when to stop training, sharing many of the weights between context classes, and smoothing context-dependent with context-independent MLPs using the approach described in Section 3.2. In addition, the Ring Array Processor (RAP) special purpose hardware, developed at ICSI (Morgan et al., 1992), allowed rapid training of such large networks on large data sets. In order to reduce the number of weights in the MLP, we are currently exploring alternative architectures which apply the smoothing techniques described here to binary context inputs.

## 6 CONCLUSIONS

MLP-based probability estimation can be useful for both improving recognition accuracy and reducing memory needs for HMM-based speech recognition systems. These benefits, however, must be weighed against increased computational requirements.

We have presented a new MLP architecture and training procedure for modeling context-dependent phonetic classes with a sequence of distributions. Tests using the DARPA Resource Management database have shown improvements in recognition performance using this new approach, modeling only generalized biphone context categories. These results suggest that sharing input-to-hidden weights between context categories (and not retraining them during the context-dependent training phase) results in a hidden layer representation which is adequate for context-dependent as well as context-independent modeling, error-based smoothing of context-independent and context-dependent weights is effective for training a robust model, and using separate output layers and hidden-to-output weights corresponding to different context classes of different states of HMM phone models is adequate to capture acoustic effects which change throughout the production of individual phonetic segments.

## Acknowledgements

The work reported here was partially supported by DARPA Contract MDA904-90-C-5253. Discussions with Herve Bourlard were very helpful.

## References

H. Bourlard, N. Morgan, C. Wooters, and S. Renals (1992), "CDNN: A Context Dependent Neural Network for Continuous Speech Recognition," *ICASSP*, pp. 349-352, San Francisco.

M. Cohen, H. Murveit, J Bernstein, P. Price, and M. Weintraub (1990), "The DECI-PHER Speech Recognition System," *ICASSP*, pp. 77-80, Alburquerque, New Mexico.

F. Jelinek and R. Mercer (1980), "Interpolated estimation of markov source parameters from sparse data," in *Pattern Recognition in Practice*, E. Gelsema and L. Kanal, Eds. Amsterdam: North-Holland, pp. 381-397.

S. Levinson, L. Rabiner, and M. Sondhi (1983), "An introduction to the application of the theory of probabilistic functions of a Markov process to automatic speech recognition," *Bell Syst. Tech. Journal 62*, pp. 1035-1074.

N. Morgan and H. Bourlard (1990), "Continuous Speech Recognition Using Multilayer Perceptrons with Hidden Markov Models," *ICASSP*, pp. 413-416, Alburquerque, New Mexico.

N. Morgan, J. Beck, P. Kohn, J. Bilmes, E. Allman, and J. Beer (1992), "The Ring Array Processor (RAP): A Multiprocessing Peripheral for Connectionist Applications," *Journal of Parallel and Distributed Computing*, pp. 248-259.

S. Renals, N. Morgan, M. Cohen, and H. Franco (1992), "Connectionist Probability Estimation in the DECIPHER Speech Recognition System," *ICASSP*, pp. 601-604, San Francisco.

D. Rumelhart, G. Hinton, and R. Williams (1986), "Learning Internal Representations by Error Propagation," in *Parallel Distributed Processing: Explorations of the Microstructure of Cognition*, vol 1: Foundations, D. Rumelhart & J. McClelland, Eds. Cambridge: MIT Press.

R. Schwartz, Y. Chow, O. Kimball, S. Roucos, M. Krasner, and J. Makhoul (1985), "Context-dependent modeling for acoustic-phonetic recognition of continuous speech," *ICASSP*, pp. 1205-1208.